# Dynamic Time-Alignment Kernel in Support Vector Machine

**Hiroshi Shimodaira**
School of Information Science,
Japan Advanced Institute of
Science and Technology
*sim@jaist.ac.jp*

**Ken-ichi Noma**
School of Information Science,
Japan Advanced Institute of
Science and Technology
*knoma@jaist.ac.jp*

**Mitsuru Nakai**
School of Information Science,
Japan Advanced Institute of
Science and Technology
*mit@jaist.ac.jp*

**Shigeki Sagayama**
Graduate School of Information Science
and Technology,
The University of Tokyo
*sagayama@hil.t.u-tokyo.ac.jp*

## Abstract

A new class of Support Vector Machine (SVM) that is applicable to sequential-pattern recognition such as speech recognition is developed by incorporating an idea of non-linear time alignment into the kernel function. Since the time-alignment operation of sequential pattern is embedded in the new kernel function, standard SVM training and classification algorithms can be employed without further modifications. The proposed SVM (DTAK-SVM) is evaluated in speaker-dependent speech recognition experiments of hand-segmented phoneme recognition. Preliminary experimental results show comparable recognition performance with hidden Markov models (HMMs).

## 1 Introduction

Support Vector Machine (SVM) [1] is one of the latest and most successful statistical pattern classifier that utilizes a kernel technique [2, 3]. The basic form of SVM classifier which classifies an input vector $\boldsymbol{x} \in \mathcal{R}^n$ is expressed as

$$g(\boldsymbol{x}) = \sum_{i=1}^{N} \alpha_i y_i \phi(\boldsymbol{x}_i) \cdot \phi(\boldsymbol{x}) + b = \sum_{i=1}^{N} \alpha_i y_i K(\boldsymbol{x}_i, \boldsymbol{x}) + b, \quad (1)$$

where $\phi$ is a non-linear mapping function $\phi(\boldsymbol{x}) : \mathcal{R}^n \mapsto \mathcal{R}^{n'}, \ (n \ll n')$, "$\cdot$" denotes the inner product operator, $\boldsymbol{x}_i, y_i$ and $\alpha_i$ are the $i$-th training sample, its class label, and its Lagrange multiplier, respectively, $K$ is a kernel function, and $b$ is a bias.

Despite the successful applications of SVM in the field of pattern recognition such as character recognition and text classification, SVM has not been applied to speech

recognition that much. This is because SVM assumes that each sample is a vector of fixed dimension, and hence it can not deal with the variable length sequences directly. Because of this, most of the efforts that have been made so far to apply SVM to speech recognition employ linear time normalization, where input feature vector sequences with different lengths are aligned to same length [4]. A variant of this approach is a hybrid of SVM and HMM (hidden Markov model), in which HMM works as a pre-processor to feed time-aligned fixed-dimensional vectors to SVM [5]. Another approach is to utilize probabilistic generative models as a SVM kernel function. This includes the Fisher kernels [6, 7], and conditional symmetric independence (CSI) kernels [8], both of which employ HMMs as the generative models. Since HMMs can treat sequential patterns, SVM that employs the generative models based on HMMs can handle sequential patterns as well.

In contrast to those approaches, our approach is a direct extension of the original SVM to the case of variable length sequence. The idea is to incorporate the operation of dynamic time alignment into the kernel function itself. Because of this, the proposed new SVM is called "Dynamic Time-Alignment Kernel SVM (DTAK-SVM)". Unlike the SVM with Fisher kernel that requires two training stages with different training criteria, one is for training the generative models and the second is for training the SVM, the DTAK-SVM uses one training criterion as well as the original SVM.

## 2  Dynamic Time-Alignment Kernel

We consider a sequence of vectors $X = (\boldsymbol{x}_1, \boldsymbol{x}_2, \cdots, \boldsymbol{x}_L)$, where $\boldsymbol{x}_i \in \mathcal{R}^n$, $L$ is the length of the sequence, and the notation $|X|$ is sometimes used to represent the length of the sequence instead. For simplification, we at first assume the so-called linear SVM that does not employ non-linear mapping function $\phi$. In such case, the kernel operation in (1) is identical to the inner product operation.

### 2.1  Formulation for linear kernel

Assume that we have two vector sequences $X$ and $V$. If these two patterns are equal in length, i.e. $|X| = |V| = L$, then the inner product between $X$ and $V$ can be obtained easily as a summation of each inner product between $\boldsymbol{x}_k$ and $\boldsymbol{v}_k$ for $k = 1, \cdots, L$:

$$X \cdot V \;\; = \;\; \sum_{k=1}^{L} \boldsymbol{x}_k \cdot \boldsymbol{v}_k, \tag{2}$$

and therefore an SVM classifier can be defined as given in (1). On the other hand in case where the two sequences are different in length, the inner product can not be calculated directly. Even in such case, however, some sort of inner product like operation can be defined if we align the lengths of the patterns. To that end, let $\psi(k), \theta(k)$ be the time-warping functions of normalized time frame $k$ for the pattern $X$ and $V$, respectively, and let "$\circ$" be the new inner product operator instead of the original inner product "$\cdot$". Then the new inner product between the two vector sequences $X$ and $V$ can be given by

$$X \circ V \;\; = \;\; \frac{1}{L} \sum_{k=1}^{L} \boldsymbol{x}_{\psi(k)} \cdot \boldsymbol{v}_{\theta(k)}, \tag{3}$$

where $L$ is a normalized length that can be either $|X|$, $|V|$ or arbitrary positive integer.

There would be two possible types of time-warping functions. One is a linear time-warping function and the other is a non-linear time-warping function. The linear time-warping function takes the form as

$$\psi(k) = \lceil (|X|/L)k \rceil, \quad \theta(k) = \lceil (|V|/L)k \rceil,$$

where $\lceil x \rceil$ is the ceiling function which gives the smallest integer that is greater than or equal to $x$. As it can be seen from the definition given above, the linear warping function is not suitable for continuous speech recognition, i.e. frame-synchronous processing, because the sequence lengths, $|X|$ and $|V|$, should be known beforehand. On the other hand, non-linear time warping, or dynamic time warping (DTW) [9] in other word, enables frame-synchronous processing. Furthermore, the past research on speech recognition has shown that the recognition performance by the non-linear time normalization outperforms the one by the linear time normalization. Because of these reasons, we focus on the non-linear time warping based on DTW.

Though the original DTW uses a distance/distortion measure and finds the optimal path that minimizes the accumulated distance/distortion, the DTW that is employed for SVM uses inner product or kernel function instead and finds the optimal path that maximizes the accumulated similarity:

$$X \circ V \quad = \quad \max_{\psi,\theta} \frac{1}{M_{\psi\theta}} \sum_{k=1}^{L} m(k) \boldsymbol{x}_{\psi(k)} \cdot \boldsymbol{v}_{\theta(k)}, \tag{4}$$

$$\text{subject to} \quad 1 \le \psi(k) \le \psi(k+1) \le |X|, \ \psi(k+1) - \psi(k) \le Q, \tag{5}$$
$$1 \le \theta(k) \le \theta(k+1) \le |V|, \ \theta(k+1) - \theta(k) \le Q,$$

where $m(k)$ is a nonnegative (path) weighting coefficient, $M_{\psi\theta}$ is a (path) normalizing factor, and $Q$ is a constant constraining the local continuity. In the standard DTW, the normalizing factor $M_{\psi\theta}$ is given as $\sum_{k=1}^{L} m(k)$, and the weighting coefficients $m(k)$ are chosen so that $M_{\psi\theta}$ is independent of the warping functions.

The above optimization problem can be solved efficiently by dynamic programming. The recursive formula in the dynamic programming employed in the present study is as follows

$$G(i,j) = \max \left\{ \begin{array}{l} G(i-1,j) + \mathrm{Inp}(i,j), \\ G(i-1,j-1) + 2\,\mathrm{Inp}(i,j), \\ G(i,j-1) + \mathrm{Inp}(i,j), \end{array} \right\} \tag{6}$$

where $\mathrm{Inp}(i,j)$ is the standard inner product between the two vectors corresponding to point $i$ and $j$. As a result, we have

$$X \circ V = G(|X|,|V|)/(|X|+|V|). \tag{7}$$

## 2.2  Formulation for non-linear kernel

In the last subsection, a linear kernel, i.e. the inner product, for two vector sequences with different lengths has been formulated in the framework of dynamic time-warping. With a little constraint, similar formulation is possible for the case where SVM's non-linear mapping function $\Phi$ is applied to the vector sequences. To that end, $\Phi$ is restricted to the one having the following form:

$$\Phi(X) = (\phi(\boldsymbol{x}_1), \phi(\boldsymbol{x}_2), \cdots, \phi(\boldsymbol{x}_L)), \tag{8}$$

where $\phi$ is a non-linear mapping function that is applied to each frame vector $\boldsymbol{x}_i$, as given in (1). It should be noted that under the above restriction $\Phi$ preserves the original length of sequence at the cost of losing long-term correlations such as the

one between $\boldsymbol{x}_1$ and $\boldsymbol{x}_L$. As a result, a new class of kernel can be defined by using the extended inner product introduced in the previous section;

$$
\mathcal{K}_s(X, V) = \Phi(X) \circ \Phi(V) \tag{9}
$$

$$
= \max_{\psi, \theta} \frac{1}{M_{\psi\theta}} \sum_{k=1}^{L} m(k)\phi(\boldsymbol{x}_{\psi(k)}) \cdot \phi(\boldsymbol{v}_{\theta(k)}) \tag{10}
$$

$$
= \max_{\psi, \theta} \frac{1}{M_{\psi\theta}} \sum_{k=1}^{L} m(k)K(\boldsymbol{x}_{\psi(k)}, \boldsymbol{v}_{\theta(k)}). \tag{11}
$$

We call this new kernel "dynamic time-alignment kernel (DTAK)".

## 2.3 Properties of the dynamic time-alignment kernel

It has not been proven that the proposed function $\mathcal{K}_s(,)$ is really an SVM's admissible kernel which guarantees the existence of a feature space. This is because that the mapping function to a feature space is not independent but dependent on the given vector sequences. Although a class of data-dependent asymmetric kernel for SVM has been developed in [10], our proposed function is more complicated and difficult to analyze because the input data is a vector sequence with variable length and non-linear time normalization is embedded in the function. Instead, what have been known about the proposed function so far are (1) $\mathcal{K}_s$ is symmetric, (2) $\mathcal{K}_s$ satisfies the Cauchy-Schwartz like inequality described bellow:

**Proposition 1**

$$
\mathcal{K}_s(X, V)^2 \leq \mathcal{K}_s(X, X)\mathcal{K}_s(V, V) \tag{12}
$$

**Proof**  For simplification, we assume that normalized length $L$ is fixed, and omit $m(k)$ and $M_{\psi\theta}$ in (11). Using the standard Cauchy-Schwartz inequality, the following inequality holds:

$$
\mathcal{K}_s(X, V) = \max_{\psi, \theta} \sum_{k=1}^{L} \phi(\boldsymbol{x}_{\psi(k)}) \cdot \phi(\boldsymbol{v}_{\theta(k)}) = \sum_{k=1}^{L} \phi(\boldsymbol{x}_{\psi^*(k)}) \cdot \phi(\boldsymbol{v}_{\theta^*(k)}) \tag{13}
$$

$$
\leq \sum_{k=1}^{L} \| \phi(\boldsymbol{x}_{\psi^*(k)}) \| \| \phi(\boldsymbol{v}_{\theta^*(k)}) \|, \tag{14}
$$

where $\psi^*(k), \theta^*(k)$ represent the optimal warping functions that maximize the RHS of (13). On the other hand,

$$
\mathcal{K}_s(X, X) = \max_{\psi, \theta} \sum_{k=1}^{L} \phi(\boldsymbol{x}_{\psi(k)}) \cdot \phi(\boldsymbol{x}_{\theta(k)}) = \sum_{k=1}^{L} \phi(\boldsymbol{x}_{\psi^+(k)}) \cdot \phi(\boldsymbol{x}_{\theta^+(k)}). \tag{15}
$$

Because here we assume that $\psi^+(k), \theta^+(k)$ are the optimal warping functions that maximize (15), for any warping functions including $\psi^*(k)$, the following inequality holds:

$$
\mathcal{K}_s(X, X) \geq \sum_{k=1}^{L} \phi(\boldsymbol{x}_{\psi^*(k)}) \cdot \phi(\boldsymbol{x}_{\psi^*(k)}) = \sum_{k=1}^{L} \| \phi(\boldsymbol{x}_{\psi^*(k)}) \|^2 . \tag{16}
$$

In the same manner, the following holds:

$$
\mathcal{K}_s(V, V) \geq \sum_{k=1}^{L} \phi(\boldsymbol{v}_{\theta^*(k)}) \cdot \phi(\boldsymbol{v}_{\theta^*(k)}) = \sum_{k=1}^{L} \| \phi(\boldsymbol{v}_{\theta^*(k)}) \|^2 . \tag{17}
$$

Therefore,

$$\mathcal{K}_s(X,X)\mathcal{K}_s(V,V) - \mathcal{K}_s(X,V)^2$$

$$\geq \left( \sum_{k=1}^{L} \| \phi(\boldsymbol{x}_{\psi^*(k)}) \|^2 \right) \left( \sum_{k=1}^{L} \| \phi(\boldsymbol{v}_{\theta^*(k)}) \|^2 \right) - \left( \sum_{k=1}^{L} \| \phi(\boldsymbol{x}_{\psi^*(k)}) \| \| \phi(\boldsymbol{v}_{\theta^*(k)}) \| \right)^2$$

$$= \sum_{i=1}^{L} \sum_{j=i+1}^{L} \left( \| \phi(\boldsymbol{x}_{\psi^*(i)}) \| \| \phi(\boldsymbol{v}_{\theta^*(j)}) \| - \| \phi(\boldsymbol{x}_{\psi^*(j)}) \| \| \phi(\boldsymbol{v}_{\theta^*(i)}) \| \right)^2 \geq 0 \quad (18)$$

∎

## 3  DTAK-SVM

Using the dynamic time-alignment kernel (DTAK) introduced in the previous section, the discriminant function of SVM for a sequential pattern is expressed as

$$g(X) = \sum_{i=1}^{N} \alpha_i y_i \Phi(X^{(i)}) \circ \Phi(X) + b \quad (19)$$

$$= \sum_{i=1}^{N} \alpha_i y_i \mathcal{K}_s(X^{(i)}, X) + b, \quad (20)$$

where $X^{(i)}$ represents the $i$-th training pattern. As it can be seen from these expressions, the SVM discriminant function for time sequence has the same form with the original SVM except for the difference in kernels. It is straightforward to deduce the learning problem which is given as

$$\min_{W,b,\xi_i} \quad \frac{1}{2}W \circ W + C \sum_{i=1}^{N} \xi_i, \quad (21)$$

$$\text{subject to} \quad y_i(W \circ \Phi(X^{(i)}) + b) \geq 1 - \xi_i, \quad (22)$$
$$\xi_i \geq 0, \quad i = 1, \cdots, N.$$

Again, since the formulation of learning problem defined above is almost the same with that for the original SVM, same training algorithms for the original SVM can be used to solve the problem.

## 4  Experiments

Speech recognition experiments were carried out to evaluated the classification performance of DTAK-SVM. As our objective is to evaluate the basic performance of the proposed method, very limited task, hand-segmented phoneme recognition task in which positions of target patterns in the utterance are known, was chosen. Continuous speech recognition task that does not require phoneme labeling would be our next step.

### 4.1  Experimental conditions

The details of the experimental conditions are given in Table 1. The training and evaluation samples were collected from the ATR speech database: A-set (5240

Table 1: Experimental conditions

|  | Experiment-1 | Experiment-2 |
|---|---|---|
| Speaker dependency | dependent | dependent |
| Phoneme classes | 6 voiced consonants | 5 vowels |
| Speakers | 5 males | 5 males and 5 females |
| Training samples | 200 samples per phoneme | 500 samples per phoneme |
| Evaluation samples | 2,035 samples in all per speaker | 2500 samples in all per speaker |
| Signal sampling | 12kHz, 10ms frame-shift | |
| Feature values | 13-MFCCs and 13-$\Delta$MFCCs | |
| Kernel type | RBF (radial basis function): $K(\boldsymbol{x}_i, \boldsymbol{x}_j) = \exp(-\frac{\|\boldsymbol{x}_i - \boldsymbol{x}_j\|^2}{\gamma^2})$ | |

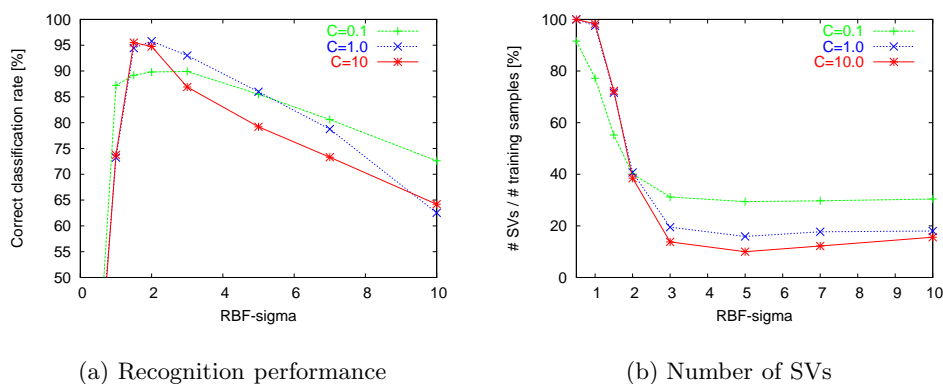

(a) Recognition performance      (b) Number of SVs

Figure 1: Experimental results for Experiment-1 (6 voiced-consonants recognition) showing (a) correct classification rate and (b) the number of SVs as a function of $\gamma$ (the parameter of RBF kernel).

Japanese words in vocabulary). In consonant-recognition task (Experiment-1), only six voiced-consonants /b,d,g,m,n,N/ were used to save time. The classification task of those 6 phonemes without using contextual information is considered as a relatively difficult task, whereas the classification of 5 vowels /a,i,u,e,o/ (Experiment-2) is considered as an easier task.

To apply SVM that is basically formulated as a two-class classifier to the multi-class problem, "one against the others" type of strategy was chosen. The proposed DTAK-SVM has been implemented with the publicly available toolkit, SVMTorch [11].

## 4.2 Experimental results

Fig. 1 depicts the experimental results for Experiment-1, where average values over 5 speakers are shown. It can be seen in Fig. 1 that the best performance of 95.8% was achieved at $\gamma = 2.0$ and $C = 10$. Similar results were obtained for Experiment-2 as given in Fig. 2.

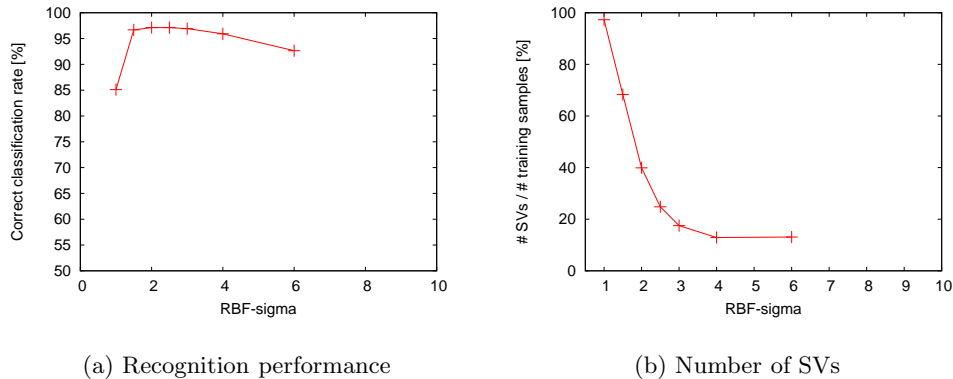

(a) Recognition performance    (b) Number of SVs

Figure 2: Experimental results for Experiment-2 (5 vowels recognition) showing (a) correct classification rate and (b) the number of SVs as a function of $\gamma$ (the parameter of RBF kernel).

Table 2: Recognition performance comparison of DTAK-SVM with HMM. Results of Experiment-1 for 1 male and 1 female speakers are shown. (numbers represent correct classification rate [%])

| Model | # training samples/phoneme | | | | | |
|---|---|---|---|---|---|---|
| | male | | | female | | |
| | 50 | 100 | 200 | 50 | 100 | 200 |
| HMM (1 mix.) | 75.0 | 69.1 | 77.1 | 72.2 | 65.5 | 76.6 |
| HMM (4 mix.) | 83.3 | 84.7 | 90.9 | 77.3 | 76.4 | 86.4 |
| HMM (8 mix.) | 82.8 | 87.0 | 92.4 | 74.6 | 79.3 | 88.5 |
| HMM (16 mix.) | 79.9 | 85.0 | 93.2 | 72.9 | 78.7 | 89.8 |
| **DTAK-SVM** | 83.8 | 85.9 | 92.1 | 83.5 | 81.8 | 87.7 |

Next, the classification performance of DTAK-SVM was compared with that of the state-of-the-art HMM. In order to see the effect of generalization performance on the size of training data set and model complexity, experiments were carried out by varying the number of training samples (50, 100, 200), and mixtures (1,4,8,16) for each state of HMM. The HMM used in this experiment was a 3-states, continuous density, Gaussian-distribution mixtures with diagonal covariances, context-independent model. HTK [12] was employed for this purpose. The parameters of DTAK-SVM were fixed to $C = 10, \gamma = 2.0$. The results for Experiment-1 with respect to 1 male and 1 female speakers are given in Table 2.

It can be said from the experimental results that DTAK-SVM shows better classification performance when the number of training samples is 50, while comparable performance when the number of samples is 200. One might argue that the number of training samples used in this experiment is not enough at all for HMM to achieve best performance. But such shortage of training samples occurs often in HMM-based real-world speech recognition, especially when context-dependent models are employed, which prevents HMM from improving the generalization performance.

# 5 Conclusions

A novel approach to extend the SVM framework for the sequential-pattern classification problem has been proposed by embedding a dynamic time-alignment operation into the kernel. Though long-term correlations between the feature vectors are omitted at the cost of achieving frame-synchronous processing for speech recognition, the proposed DTAK-SVMs demonstrated comparable performance in hand-segmented phoneme recognition with HMMs. The DTAK-SVM is potentially applicable to continuous speech recognition with some extension of One-pass search algorithm [9].

# References

[1] V. N. Vapnik, *Statistical Learning Theory*. Wiley, 1998.

[2] B. Schölkopf, C. J. Burges, and A. J. Smola, eds., *Advances in Kernel Methods*. The MIT Press, 1998.

[3] "Kernel machine website," 2000. http://www.kernel-machines.org/.

[4] P. Clarkson, "On the Use of Support Vector Machines for Phonetic Classification," in *ICASSP99*, pp. 585–588, 1999.

[5] A. Ganapathiraju and J. Picone, "Hybrid SVM/HMM architectures for speech recognition," in *ICSLP2000*, 2000.

[6] Tommi S. Jaakkola and David Haussler, "Exploiting generative models in discriminative classifiers," in *Advances in Neural Information Processing Systems 11* (M. S. Kearns and S. A. Solla and D. A. Cohn, ed.), pp. 487–493, The MIT Press, 1999.

[7] N. Smith and M. Niranjan, "Data-dependent Kernels in SVM classification of speech patterns," in *ICSLP-2000*, vol. 1, pp. 297–300, 2000.

[8] C. Watkins, "Dynamic Alignment Kernels," in *Advances in Large Margin Classifiers* (A. J. Smola and P. L. Bartlett and B. Schölkopf and D. Schuurmans, ed.), ch. 3, pp. 39–50, The MIT Press, 2000.

[9] L. Rabiner and B. Juang, *Fundamental of Speech Recognition*. Prentice Hall, 1993.

[10] K. Tsuda, "Support Vector Classifier with Asymmetric Kernel Functions," in *European Symposium on Artificial Neural Networks (ESANN)*, pp. 183–188, 1999.

[11] R. Collobert, "SVMTorch: A Support Vector Machine for Large-Scale Regression and Classification Problems," 2000. http://www.idiap.ch/learning/SVMTorch.html.

[12] "The Hidden Markov Model Toolkit (HTK)." http://htk.eng.cam.ac.uk/.
